# Conditional Random Fields for Object Recognition

**Ariadna Quattoni    Michael Collins    Trevor Darrell**
MIT Computer Science and Artificial Intelligence Laboratory
Cambridge, MA 02139
{ariadna, mcollins, trevor}@csail.mit.edu

## Abstract

We present a discriminative part-based approach for the recognition of object classes from unsegmented cluttered scenes. Objects are modeled as flexible constellations of parts conditioned on local observations found by an interest operator. For each object class the probability of a given assignment of parts to local features is modeled by a Conditional Random Field (CRF). We propose an extension of the CRF framework that incorporates hidden variables and combines class conditional CRFs into a unified framework for part-based object recognition. The parameters of the CRF are estimated in a maximum likelihood framework and recognition proceeds by finding the most likely class under our model. The main advantage of the proposed CRF framework is that it allows us to relax the assumption of conditional independence of the observed data (i.e. local features) often used in generative approaches, an assumption that might be too restrictive for a considerable number of object classes.

## 1   Introduction

The problem that we address in this paper is that of learning object categories from supervised data. Given a training set of $n$ pairs $(\mathbf{x}_i, y_i)$, where $\mathbf{x}_i$ is the ith image and $y_i$ is the category of the object present in $\mathbf{x}_i$, we would like to learn a model that maps images to object categories. In particular, we are interested in learning to recognize rigid objects such as cars, motorbikes, and faces from one or more fixed view-points.

The part-based models we consider represent images as sets of *patches*, or local features, which are detected by an interest operator such as that described in [4]. Thus an image $\mathbf{x}_i$ can be considered to be a vector $\{x_{i,1}, \ldots, x_{i,m}\}$ of $m$ patches. Each patch $x_{i,j}$ has a feature-vector representation $\phi(x_{i,j}) \in \mathbb{R}^d$; the feature vector might capture various features of the appearance of a patch, as well as features of its relative location and scale. This scenario presents an interesting challenge to conventional classification approaches in machine learning, as the input space $\mathbf{x}_i$ is naturally represented as a set of feature-vectors $\{\phi(x_{i,1}), \ldots, \phi(x_{i,m})\}$ rather than as a single feature vector. Moreover, the local patches underlying the local feature vectors may have complex interdependencies: for example, they may correspond to different parts of an object, whose spatial arrangement is important to the classification task.

The most widely used approach for part-based object recognition is the generative model proposed in [1]. This classification system models the appearance, spatial relations and co-occurrence of local parts. One limitation of this framework is that to make the model

computationally tractable one has to assume the independence of the observed data (i.e., local features) given their assignment to parts in the model. This assumption might be too restrictive for a considerable number of object classes made of structured patterns.

A second limitation of generative approaches is that they require a model $P(x_{i,j}|h_{i,j})$ of patches $x_{i,j}$ given underlying variables $h_{i,j}$ (e.g., $h_{i,j}$ may be a hidden variable in the model, or may simply be $y_i$). Accurately specifying such a generative model may be challenging – in particular in cases where patches overlap one another, or where we wish to allow a hidden variable $h_{i,j}$ to depend on several surrounding patches. A more direct approach may be to use a feature vector representation of patches, and to use a discriminative learning approach. We follow an approach of this type in this paper.

Similar observations concerning the limitations of generative models have been made in the context of natural language processing, in particular in sequence-labeling tasks such as part-of-speech tagging [7, 5, 3] and in previous work on conditional random fields (CRFs) for vision [2]. In sequence-labeling problems for NLP each observation $x_{i,j}$ is typically the $j$'th word for some input sentence, and $h_{i,j}$ is a hidden state, for example representing the part-of-speech of that word. Hidden Markov models (HMMs), a generative approach, require a model of $P(x_{i,j}|h_{i,j})$, and this can be a challenging task when features such as word prefixes or suffixes are included in the model, or where $h_{i,j}$ is required to depend directly on words other than $x_{i,j}$. This has led to research on discriminative models for sequence labeling such as MEMM's [7, 5] and conditional random fields (CRFs)[3]. A strong argument for these models as opposed to HMMs concerns their flexibility in terms of representation, in that they can incorporate essentially arbitrary feature-vector representations $\phi(x_{i,j})$ of the observed data points.

We propose a new model for object recognition based on Conditional Random Fields. We model the conditional distribution $p(y|\mathbf{x})$ directly. A key difference of our approach from previous work on CRFs is that we make use of hidden variables in the model. In previous work on CRFs (e.g., [2, 3]) each "label" $y_i$ is a sequence $\mathbf{h}_i = \{h_{i,1}, h_{i,2}, \ldots, h_{i,m}\}$ of labels $h_{i,j}$ for each observation $x_{i,j}$. The label sequences are typically taken to be fully observed on training examples. In our case the labels $y_i$ are unstructured labels from some fixed set of object categories, and the relationship between $y_i$ and each observation $x_{i,j}$ is not clearly defined. Instead, we model intermediate part-labels $h_{i,j}$ as hidden variables in the model. The model defines conditional probabilities $P(y, \mathbf{h} \mid \mathbf{x})$, and hence indirectly $P(y \mid \mathbf{x}) = \sum_{\mathbf{h}} P(y, \mathbf{h} \mid \mathbf{x})$, using a CRF. Dependencies between the hidden variables $\mathbf{h}$ are modeled by an undirected graph over these variables. The result is a model where inference and parameter estimation can be carried out using standard graphical model algorithms such as belief propagation [6].

## 2  The Model

### 2.1  Conditional Random Fields with Hidden Variables

Our task is to learn a mapping from images $\mathbf{x}$ to labels $y$. Each $y$ is a member of a set $\mathcal{Y}$ of possible image labels, for example, $\mathcal{Y} = \{\texttt{background}, \texttt{car}\}$. We take each image $\mathbf{x}$ to be a vector of $m$ "patches" $\mathbf{x} = \{x_1, x_2, \ldots, x_m\}$.[1] Each patch $x_j$ is represented by a feature vector $\phi(x_j) \in \mathbb{R}^d$. For example, in our experiments each $x_j$ corresponds to a patch that is detected by the feature detector in [4]; section [3] gives details of the feature-vector representation $\phi(x_j)$ for each patch. Our training set consists of labeled images $(\mathbf{x}_i, y_i)$ for $i = 1 \ldots n$, where each $y_i \in \mathcal{Y}$, and each $\mathbf{x}_i = \{x_{i,1}, x_{i,2}, \ldots, x_{i,m}\}$. For any image $\mathbf{x}$ we also assume a vector of "parts" variables $\mathbf{h} = \{h_1, h_2, \ldots, h_m\}$. These variables are not observed on training examples, and will therefore form a set of hidden variables in the

model. Each $h_j$ is a member of $\mathcal{H}$ where $\mathcal{H}$ is a finite set of possible parts in the model. Intuitively, each $h_j$ corresponds to a labeling of $x_j$ with some member of $\mathcal{H}$. Given these definitions of image-labels $y$, images $\mathbf{x}$, and part-labels $\mathbf{h}$, we will define a conditional probabilistic model:

$$P(y, \mathbf{h} \mid \mathbf{x}, \theta) = \frac{e^{\Psi(y, \mathbf{h}, \mathbf{x}; \theta)}}{\sum_{y'} \sum_{\mathbf{h}} e^{\Psi(y', \mathbf{h}, \mathbf{x}; \theta)}}. \tag{1}$$

Here $\theta$ are the parameters of the model, and $\Psi(y, \mathbf{h}, \mathbf{x}; \theta) \in \mathbb{R}$ is a potential function parameterized by $\theta$. We will discuss the choice of $\Psi$ shortly. It follows that

$$P(y \mid \mathbf{x}, \theta) = \sum_{\mathbf{h}} P(y, \mathbf{h} \mid \mathbf{x}, \theta) = \frac{\sum_{\mathbf{h}} e^{\Psi(y, \mathbf{h}, \mathbf{x}; \theta)}}{\sum_{y', \mathbf{h}} e^{\Psi(y', \mathbf{h}, \mathbf{x}; \theta)}}. \tag{2}$$

Given a new test image $\mathbf{x}$, and parameter values $\theta^*$ induced from a training example, we will take the label for the image to be $\arg\max_{y \in \mathcal{Y}} P(y \mid \mathbf{x}, \theta^*)$. Following previous work on CRFs [2, 3], we use the following objective function in training the parameters:

$$L(\theta) = \sum_{i} \log P(y_i \mid \mathbf{x}_i, \theta) - \frac{1}{2\sigma^2} ||\theta||^2 \tag{3}$$

The first term in Eq. 3 is the log-likelihood of the data. The second term is the log of a Gaussian prior with variance $\sigma^2$, i.e., $P(\theta) \sim \exp\left(\frac{1}{2\sigma^2} ||\theta||^2\right)$. We will use gradient ascent to search for the optimal parameter values, $\theta^* = \arg\max_\theta L(\theta)$, under this criterion.

We now turn to the definition of the potential function $\Psi(y, \mathbf{h}, \mathbf{x}; \theta)$. We assume an undirected graph structure, with the hidden variables $\{h_1, \ldots, h_m\}$ corresponding to vertices in the graph. We use $E$ to denote the set of edges in the graph, and we will write $(j, k) \in E$ to signify that there is an edge in the graph between variables $h_j$ and $h_k$. In this paper we assume that $E$ is a tree.[2] We define $\Psi$ to take the following form:

$$\Psi(y, \mathbf{h}, \mathbf{x}; \theta) = \sum_{j=1}^{m} \sum_{l} f_l^1(j, y, h_j, \mathbf{x})\theta_l^1 + \sum_{(j,k) \in E} \sum_{l} f_l^2(j, k, y, h_j, h_k, \mathbf{x})\theta_l^2 \tag{4}$$

where $f_l^1, f_l^2$ are functions defining the features in the model, and $\theta_l^1, \theta_l^2$ are the components of $\theta$. The $f^1$ features depend on single hidden variable values in the model, the $f^2$ features can depend on pairs of values. Note that $\Psi$ is linear in the parameters $\theta$, and the model in Eq. 1 is a log-linear model. Moreover the features respect the structure of the graph, in that no feature depends on more than two hidden variables $h_j, h_k$, and if a feature does depend on variables $h_j$ and $h_k$ there must be an edge $(j, k)$ in the graph $E$.

Assuming that the edges in $E$ form a tree, and that $\Psi$ takes the form in Eq. 4, then exact methods exist for inference and parameter estimation in the model. This follows because belief propagation [6] can be used to calculate the following quantities in $O(|E||\mathcal{Y}|)$ time:

$$\forall y \in \mathcal{Y}, \quad Z(y \mid \mathbf{x}, \theta) = \sum_{\mathbf{h}} \exp\{\Psi(y, \mathbf{h}, \mathbf{x}; \theta)\}$$

$$\forall y \in \mathcal{Y}, j \in 1 \ldots m, a \in \mathcal{H}, \quad P(h_j = a \mid y, \mathbf{x}, \theta) = \sum_{\mathbf{h}: h_j = a} P(\mathbf{h} \mid y, \mathbf{x}, \theta)$$

$$\forall y \in \mathcal{Y}, (j, k) \in E, a, b \in \mathcal{H}, \quad P(h_j = a, h_k = b \mid y, \mathbf{x}, \theta) = \sum_{\mathbf{h}: h_j = a, h_k = b} P(\mathbf{h} \mid y, \mathbf{x}, \theta)$$

The first term $Z(y \mid \mathbf{x}, \theta)$ is a partition function defined by a summation over the $\mathbf{h}$ variables. Terms of this form can be used to calculate $P(y \mid \mathbf{x}, \theta) = Z(y \mid \mathbf{x}, \theta) / \sum_{y'} Z(y' \mid \mathbf{x}, \theta)$. Hence inference—calculation of $\arg \max P(y \mid \mathbf{x}, \theta)$— can be performed efficiently in the model. The second and third terms are marginal distributions over individual variables $h_j$ or pairs of variables $h_j, h_k$ corresponding to edges in the graph. The next section shows that the gradient of $L(\theta)$ can be defined in terms of these marginals, and hence can be calculated efficiently.

## 2.2 Parameter Estimation Using Belief Propagation

This section considers estimation of the parameters $\theta^* = \arg \max L(\theta)$ from a training sample, where $L(\theta)$ is defined in Eq. 3. In our work we used a conjugate-gradient method to optimize $L(\theta)$ (note that due to the use of hidden variables, $L(\theta)$ has multiple local minima, and our method is therefore not guaranteed to reach the globally optimal point). In this section we describe how the gradient of $L(\theta)$ can be calculated efficiently. Consider the likelihood term that is contributed by the $i$'th training example, defined as:

$$L_i(\theta) = \log P(y_i \mid \mathbf{x}_i, \theta) = \log \left( \frac{\sum_{\mathbf{h}} e^{\Psi(y_i, \mathbf{h}, \mathbf{x}_i; \theta)}}{\sum_{y', \mathbf{h}} e^{\Psi(y', \mathbf{h}, \mathbf{x}_i; \theta)}} \cdot \right) \tag{5}$$

We first consider derivatives with respect to the parameters $\theta_l^1$ corresponding to features $f_l^1(j, y, h_j, \mathbf{x})$ that depend on single hidden variables. Taking derivatives gives

$$\frac{\partial L_i(\theta)}{\partial \theta_l^1} = \sum_{\mathbf{h}} P(\mathbf{h} \mid y_i, \mathbf{x}_i, \theta) \frac{\partial \Psi(y_i, \mathbf{h}, \mathbf{x}_i; \theta)}{\partial \theta_l^1} - \sum_{y', \mathbf{h}} P(y', \mathbf{h} \mid \mathbf{x}_i, \theta) \frac{\partial \Psi(y', \mathbf{h}, \mathbf{x}_i; \theta)}{\partial \theta_l^1}$$

$$= \sum_{\mathbf{h}} P(\mathbf{h} \mid y_i, \mathbf{x}_i, \theta) \sum_{j=1}^{m} f_l^1(j, y_i, h_j, \mathbf{x}_i) - \sum_{y', \mathbf{h}} P(y', \mathbf{h} \mid \mathbf{x}_i, \theta) \sum_{j=1}^{m} f_l^1(j, y', h_j, \mathbf{x}_i)$$

$$= \sum_{j,a} P(h_j = a \mid y_i, \mathbf{x}_i, \theta) f_l^1(j, y_i, a, \mathbf{x}_i) - \sum_{y', j, a} P(h_j = a, y' \mid \mathbf{x}_i, \theta) f_l^1(j, y', a, \mathbf{x}_i)$$

It follows that $\frac{\partial L_i(\theta)}{\partial \theta_l^1}$ can be expressed in terms of components $P(h_j = a \mid \mathbf{x}_i, \theta)$ and $P(y \mid \mathbf{x}_i, \theta)$, which can be calculated using belief propagation, provided that the graph $E$ forms a tree structure. A similar calculation gives

$$\frac{\partial L_i(\theta)}{\partial \theta_l^2} = \sum_{(j,k) \in E, a, b} P(h_j = a, h_k = b \mid y_i, \mathbf{x}_i, \theta) f_l^2(j, k, y_i, a, b, \mathbf{x}_i)$$

$$- \sum_{y', (j,k) \in E, a, b} P(h_j = a, h_k = b, y' \mid \mathbf{x}_i, \theta) f_l^2(j, k, y', a, b, \mathbf{x}_i)$$

hence $\partial L_i(\theta) / \partial \theta_l^2$ can also be expressed in terms of expressions that can be calculated using belief propagation.

## 2.3 The Specific Form of our Model

We now turn to the specific form for the model in this paper. We define

$$\Psi(y, \mathbf{h}, \mathbf{x}; \theta) = \sum_j \phi(x_j) \cdot \theta(h_j) + \sum_j \theta(y, h_j) + \sum_{(j,k) \in E} \theta(y, h_j, h_k) \tag{6}$$

Here $\theta(k) \in \mathbb{R}^d$ for $k \in \mathcal{H}$ is a parameter vector corresponding to the $k$'th part label. The inner-product $\phi(x_j) \cdot \theta(h_j)$ can be interpreted as a measure of the compatibility between patch $x_j$ and part-label $h_j$. Each parameter $\theta(y, k) \in \mathbb{R}$ for $k \in \mathcal{H}, y \in \mathcal{Y}$ can be

interpreted as a measure of the compatibility between part $k$ and label $y$. Finally, each parameter $\theta(y, k, l) \in \mathbb{R}$ for $y \in \mathcal{Y}$, and $k, l \in \mathcal{H}$ measures the compatibility between an edge with labels $k$ and $l$ and the label $y$. It is straightforward to verify that the definition in Eq. 6 can be written in the same form as Eq. 4. Hence belief propagation can be used for inference and parameter estimation in the model.

The patches $x_{i,j}$ in each image are obtained using the SIFT detector [4]. Each patch $x_{i,j}$ is then represented by a feature vector $\phi(x_{i,j})$ that incorporates a combination of SIFT and relative location and scale features.

The tree $E$ is formed by running a minimum spanning tree algorithm over the parts $h_{i,j}$, where the cost of an edge in the graph between $h_{i,j}$ and $h_{i,k}$ is taken to be the distance between $x_{i,j}$ and $x_{i,k}$ in the image. Note that the structure of $E$ will vary across different images. Our choice of $E$ encodes our assumption that parts conditioned on features that are spatially close are more likely to be dependent. In the future we plan to experiment with the minimum spanning tree approach under other definitions of edge-cost. We also plan to investigate more complex graph structures that involve cycles, which may require approximate methods such as loopy belief propagation for parameter estimation and inference.

## 3 Experiments

We carried out three sets of experiments on a number of different data sets.[3] The first two experiments consisted of training a two class model (object vs. background) to distinguish between a category from a single viewpoint and background. The third experiment consisted of training a multi-class model to distinguish between $n$ classes.

The only parameter that was adjusted in the experiments was the scale of the images upon which the interest point detector was run. In particular, we adjusted the scale on the car side data set: in this data set the images were too small and without this adjustment the detector would fail to find a significant amount of features.

For the experiments we randomly split each data set into three separate data sets: training, validation and testing. We use the validation data set to set the variance parameters $\sigma^2$ of the gaussian prior.

### 3.1 Results

In figure 2.a we show how the number of parts in the model affects performance. In the case of the car side data set, the ten-part model shows a significant improvement compared to the five parts model while for the car rear data set the performance improvement obtained by increasing the number of parts is not as significant. Figure 2.b shows a performance comparison with previous approaches [1] tested on the same data set (though on a different partition). We observe an improvement between 2 % and 5 % for all data sets.

Figures 3 and 4 show results for the multi-class experiments. Notice that random performance for the animal data set would be 25 % across the diagonal. The model exhibits best performance for the Leopard data set, for which the presence of part 1 alone is a clear predictor of the class. This shows again that our model can learn discriminative part distributions for each class. Figure 3 shows results for a multi-view experiment where the task is two distinguish between two different views of a car and background.

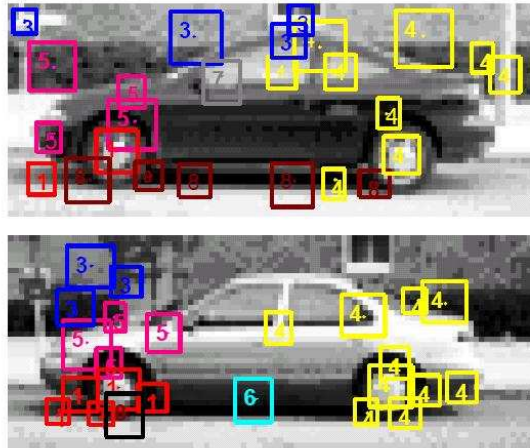

Figure 1: Examples of the most likely assignment of parts to features for the two class experiments (car data set).

(a)

| Data set | 5 parts | 10 parts |
|----------|---------|----------|
| Car Side | 94 % | 99 % |
| Car Rear | 91 % | 91.7 % |

(b)

| Data set | Our Model | Others [1] |
|----------|-----------|------------|
| Car Side | 99 % | - |
| Car Rear | 94.6 % | 90.3 % |
| Face | 99 % | 96.4 % |
| Plane | 96 % | 90.2 % |
| Motorbike | 95 % | 92.5 % |

Figure 2: (a) Equal Error Rates for the car side and car rear experiments with different number of parts. (b) Comparative Equal Error Rates.

Figure 1 displays the Viterbi labeling[4] for a set of example images showing the most likely assignment of local features to parts in the model. Figure 6 shows the mean and variance of each part's location for car side images and background images. The mean and variance of each part's location for the car side images were calculated in the following manner: First we find for every image classified as class $a$ the most likely part assignment under our model. Second, we calculate the mean and variance of positions of all local features that were assigned to the same part. Similarly Figure 5 shows part counts among the Viterbi paths assigned to examples of a given class.

As can be seen in Figure 6 , while the mean location of a given part in the background images and the mean location of the same part in the car images are very similar, the parts in the car have a much tighter distribution which seems to suggest that the model is learning the shape of the object.

As shown in Figure 5 the model has also learnt discriminative part distributions for each class, for example the presence of part 1 seems to be a clear predictor for the car class. In general part assignments seem to rely on a combination of appearance and relative location. Part 1, for example, is assigned to wheel like patterns located on the left of the object.

| Data set | Precision | Recall |
|---|---|---|
| Car Side | 87.5 % | 98 % |
| Car Rear | 87.4 % | 86.5 % |

Figure 3: Precision and recall results for 3 class experiment.

| Data set | Leopards | Llamas | Rhinos | Pigeons |
|---|---|---|---|---|
| Leopards | 91 % | 2 % | 0 % | 7 % |
| Llamas | 0 % | 50 % | 27 % | 23 % |
| Rhinos | 0 % | 40 % | 46 % | 14 % |
| Pigeons | 0 % | 30 % | 20 % | 50 % |

Figure 4: Confusion table for 4 class experiment.

However, the parts might not carry semantic meaning. It appears that the model has learnt a vocabulary of very general parts with significant variability in appearance and learns to discriminate between classes by capturing the most likely arrangement of these parts for each class.

In some cases the model relies more heavily on relative location than appearance because the appearance information might not be very useful for discriminating between the two classes. One of the reasons for this is that the detector produces a large number of false detections, making the appearance data too noisy for discrimination. The fact that the model is able to cope with this lack of discriminating appearance information illustrates its flexible data-driven nature. This can be a desirable model property of a general object recognition system, because for some object classes appearance is the important discriminant (i.e., in textured classes) while for others shape may be important (i.e., in geometrically constrained classes).

One noticeable difference between our model and similar part-based models is that our model learns large parts composed of small local features. This is not surprising given how the part dependencies were built (i.e., through their position in minimum spanning tree): the potential functions defined on pairs of hidden variables tend to smooth the allocation of parts to patches.

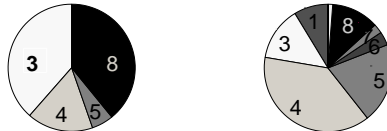

Figure 5: Graph showing part counts for the background (left) and car side images (right)

## 4   Conclusions and Further Work

In this work we have presented a novel approach that extends the CRF framework by incorporating hidden variables and combining class conditional CRFs into an unified framework for object recognition. Similarly to CRFs and other maximum entropy models our approach allows us to combine arbitrary observation features for training discriminative classifiers with hidden variables. Furthermore, by making some assumptions about the joint distribution of hidden variables one can derive efficient training algorithms based on dynamic programming.

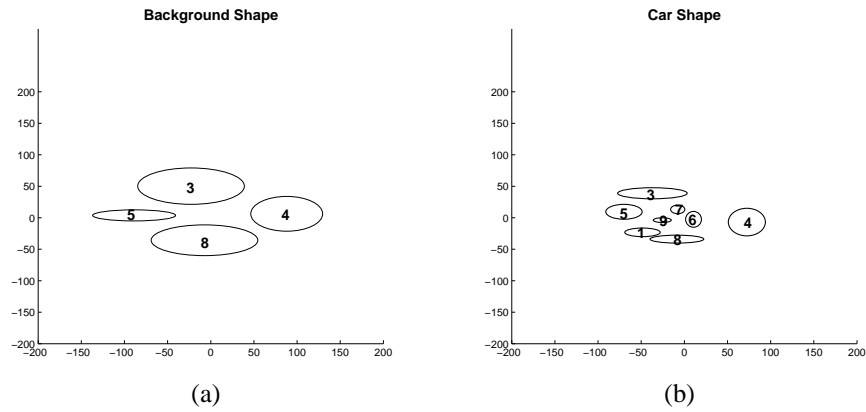

Figure 6: (a) Graph showing mean and variance of locations for the different parts for the car side images; (b) Mean and variance of part locations for the background images.

The main limitation of our model is that it is dependent on the feature detector picking up discriminative features of the object. Furthermore, our model might learn to discriminate between classes based on the statistics of the feature detector and not the true underlying data, to which it has no access. This is not a desirable property since it assumes the feature detector to be consistent. As future work we would like to incorporate the feature detection process into the model.

## Footnotes

[1]Note that the number of patches $m$ can vary across images, and did vary in our experiments. For convenience we use notation where $m$ is fixed across different images; in reality it will vary across images but this leads to minor changes to the model.

[2]This will allow exact methods for inference and parameter estimation in the model, for example using belief propagation. If $E$ contains cycles then approximate methods, such as loopy belief-propagation, may be necessary for inference and parameter estimation.

[3]The images were obtained from http://www.vision.caltech.edu/html-files/archive.html and the car side images from http://l2r.cs.uiuc.edu/ cogcomp/Data/Car/. Notice, that since our algorithm does not currently allow for the recognition of multiple instances of an object we test it on a partition of the the training set in http://l2r.cs.uiuc.edu/ cogcomp/Data/Car/ and not on the testing set in that site. The animals data set is a subset of Caltech's 101 categories data set.

[4]This is the labeling $\mathbf{h}^* = \arg\max_{\mathbf{h}} P(\mathbf{h} \mid y, \mathbf{x}, \theta)$ where $\mathbf{x}$ is an image and $y$ is the label for the image under the model.

# References

[1] R. Fergus, P. Perona,and A. Zisserman. Object class recognition by unsupervised scale-invariant learning. In *Proceedings of the IEEE Conference on Computer Vision and Pattern Recognition*,volume 2, pages 264-271, June 2003.

[2] S. Kumar and M. Hebert. Discriminative random fields: A framework for contextual interaction in classification. In *IEEE Int Conference on Computer Vision*,volume 2, pages 1150-1157, June 2003.

[3] J. Lafferty,A. McCallum and F. Pereira. Conditional random fields: Probabilistic models for segmenting and labeling sequence data. In *Proc. Int Conf. on Machine Learning*, 2001.

[4] D. Lowe. Object Recognition from local scale-invariant features. In *IEEE Int Conference on Computer Vision*, 1999.

[5] A. McCallum, D. Freitag, and F. Pereira. Maximum entropy markov models for information extraction and segmentation. In *ICML-2000*, 2000.

[6] J. Pearl. *Probabilistic Reasoning in Intelligent Systems: Networks of Plausible Inference*. Morgan Kaufmann,1988.

[7] A. Ratnaparkhi. A maximum entropy part-of-speech tagger. In *EMNLP*, 1996.
